# A Hybrid Radial Basis Function Neurocomputer and Its Applications

**Steven S. Watkins**
ECE Department
UCSD
La Jolla, CA. 92093

**Paul M. Chau**
ECE Department
UCSD
La Jolla, CA. 92093

**Raoul Tawel**
JPL
Caltech
Pasadena, CA. 91109

**Bjorn Lambrigtsen**
JPL
Caltech
Pasadena, CA. 91109

**Mark Plutowski**
CSE Department
UCSD
La Jolla, CA. 92093

## Abstract

A neurocomputer was implemented using radial basis functions and a combination of analog and digital VLSI circuits. The hybrid system uses custom analog circuits for the input layer and a digital signal processing board for the hidden and output layers. The system combines the advantages of both analog and digital circuits, featuring low power consumption while minimizing overall system error. The analog circuits have been fabricated and tested, the system has been built, and several applications have been executed on the system. One application provides significantly better results for a remote sensing problem than have been previously obtained using conventional methods.

## 1.0 Introduction

This paper describes a neurocomputer development system that uses a radial basis function as the transfer function of a neuron rather than the traditional sigmoid function. This neurocomputer is a hybrid system which has been implemented with a combination of analog and digital VLSI technologies. It offers the low-power advantage of analog circuits operating in the subthreshold region and the high-precision advantage of digital circuits. The system is targeted for applications that require low-power operation and use input data in analog form, particularly remote sensing and portable computing applications. It has already provided significantly better results for a remote sensing

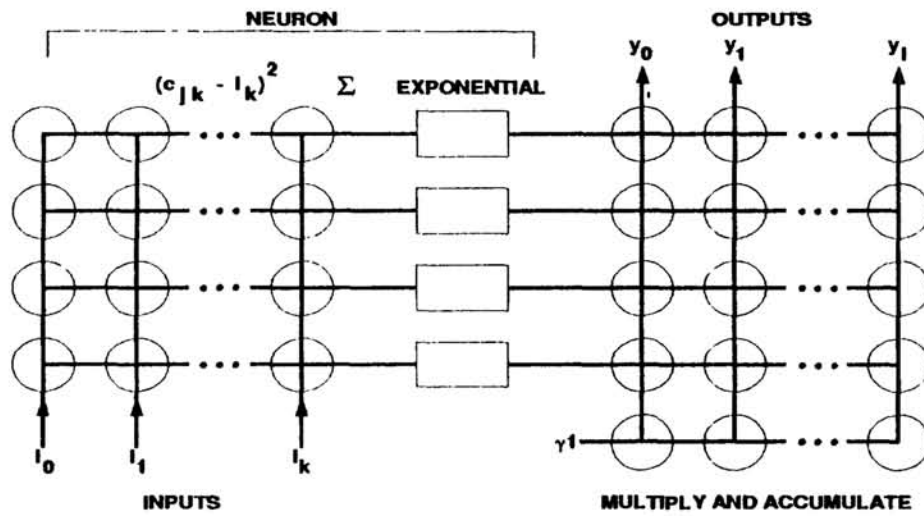

Figure 1: Radial Basis Function Network

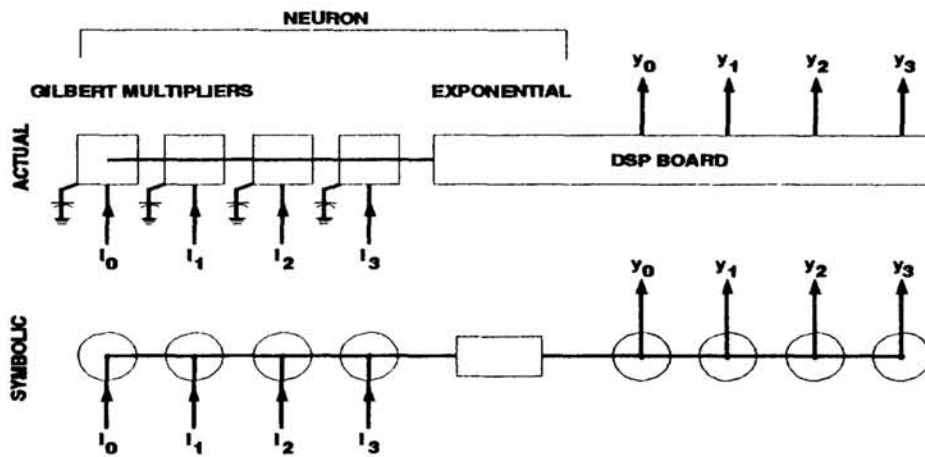

Figure 2: Mapping of RBF Network to Hardware

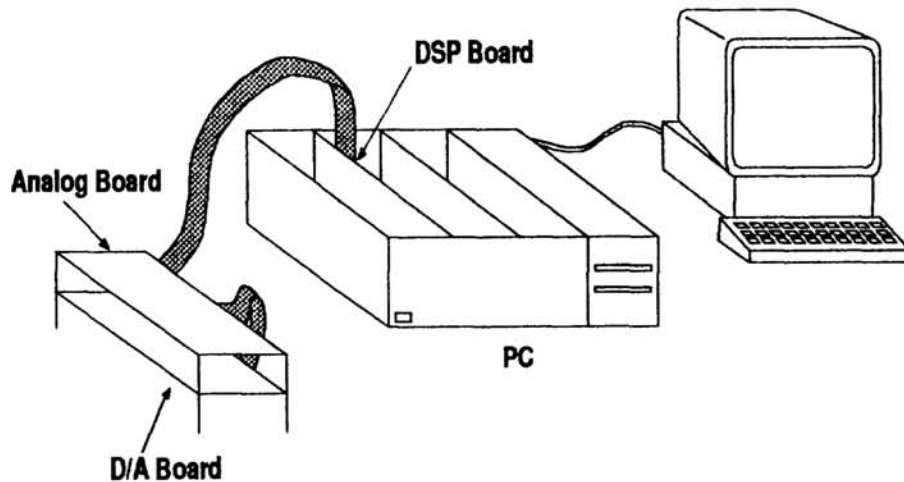

Figure 3: The RBF Neurocomputer Development System

climate problem than have been previously obtained using conventional methods.

Figure 1 illustrates a radial basis function (RBF) network. Radial basis functions have been used to solve mapping and function estimation problems with positive results (Moody and Darken, 1989; Lippman, 1991). When coupled with a dynamic neuron allocation algorithm such as Platt's RANN (Platt, 1991), RBF networks can usually be trained much more quickly than a traditional sigmoidal, back-propagation network.

RBF networks have been implemented with completely-analog (Platt, Anderson and Kirk, 1993), completely-digital (Watkins, Chau and Tawel, Nov., 1992), and with hybrid analog/digital approaches (Watkins, Chau and Tawel, Oct., 1992). The hybrid approach is optimal for applications which require low power consumption and use input data that is naturally in the analog domain while also requiring the high precision of the digital domain.

## 2.0 System Architecture and Benefits

Figure 2 shows the mapping of the RBF network to hardware. Figure 3 shows the neurocomputer development system. The system consists of a PC controller, a DSP board with a Motorola 56000 DSP chip and a board with analog multipliers. The benefits of the hybrid approach are lower-cost parallelism than is possible with a completely-digital system, and more precise computation than is possible with a completely-analog system.

The parallelism is available for low cost in terms of area and power, when the inputs are in the analog domain. When comparing a single analog multiplier to a 10-bit fixed point digital multiplier, the analog cell uses less than one-quarter the area and approximately five orders of magnitude less power. When comparing an array of analog multipliers to a Motorola 56000 DSP chip, 1000 Gilbert multipliers can fit in an area about half the size of the DSP chip, while consuming .003% of the power.

The restriction of requiring analog inputs is placed on the system, because if the inputs were digital, the high cost of D to A conversion would remove the low cost benefit of the system. This restriction causes the neurocomputer to be targeted for applications using inputs that are in the analog domain, such as remote sensing applications that use microwave or infrared sensors and speech recognition applications that use analog filters.

The hybrid system reduces the overall system error when compared with a completely-analog solution. The digital circuits compute the hidden and output layers with 24 bits of precision while analog circuits are limited to about 8 bits of precision. Also the RANN algorithm requires a large range of width variation for the Gaussian function and this is more easily achieved with digital computation. Completely analog solutions to this problem are severely limited by the voltage rails of the chip.

## 3.0 Circuits

Several different analog circuit approaches were explored as possible implementations of the network. After the dust settled, we chose to implement only the input layer with analog circuits because it offers the greatest opportunity for parallelism, providing parallel performance benefits at a low cost in terms of area and power. The input layer requires more than $O(N^2)$ computations (where N is the number of neurons), while the hidden and output layers require only $O(N)$ computations (because there is one hidden layer computation per neuron and the number of outputs is either one or very small).

The analog circuits used in the input layer are Gilbert multipliers (Mead, 1989). The circuits were fabricated with 2.0 micron, double-poly, P-well, CMOS technology. The Gilbert cell performs the operation of multiplying two voltage differences: $(V1-V2)x(V3-V4)$. In this system, $V1=V3$ and $V2=V4$, which causes the circuit to compute the square of the difference between a stored weight and the input. The current outputs of the Gilbert cells in a row are wired together to sum their currents, giving a sum of squared errors. This current is converted to a voltage, fed to an A to D converter and then passed to the DSP board where the hidden and output layers are computed. The radial basis function (Gaussian) of the hidden layer is computed by using a lookup table. The system uses the fast multiply/accumulate operation of the DSP chip to compute the output layer.

## 4.0 Applications

The low-power feature of the hybrid system makes it attractive for applications where power consumption is a prime consideration, such as satellite-based applications and portable computing (using battery power). The neurocomputer has been applied to three problems: a remote sensing climate problem, the Mackey-Glass chaotic time series estimation and speech phoneme recognition. The remote sensing application falls into the satellite category. The Mackey-Glass and speech recognition applications are potentially portable. Systems for these applications are likely to have inputs in the analog domain (eliminating the need for D to A conversion, as already discussed) making it feasible to execute them on the hybrid neurocomputer.

### 4.1 The Remote Sensing Application

The remote sensing problem is an inverse mapping problem that uses microwave energy in different bands as input to predict the water vapor content of the atmosphere at different altitudes. Water vapor content is a key parameter for predicting weather in the tropics and mid-latitudes (Kakar and Lambrigtsen, 1984). The application uses 12 inputs and 1 output. The system input is naturally in analog form, the result of amplified microwave signals, so no D to A conversion of input data is required. Others have used neural networks with success to perform a similar inverse mapping to predict the temperature gradient of the atmosphere (Motteler et al., 1993). Section 5 details the improved results of the RBF network over conventional methods. Since water vapor content is a very important component of climate models, improved results in predicted water vapor content means improved climate models.

Remote sensing problems require satellite hardware where power consumption is always a major constraint. The low-power nature of the hybrid network would allow the network to be placed on board a satellite. With future EOS missions requiring several thousand sensors, the on-board network would reduce the bandwidth requirements of the data being sent back to earth, allowing the reduced water vapor content data to be transmitted rather than the raw sensor data. This data bandwidth reduction could be used either to send back more meaningful data to further improve climate models, or to reduce the amount of data transmitted, saving energy.

### 4.2 The Mackey-Glass Application

The Mackey-Glass chaotic time series application uses several previous time sample values to predict the current value of a time series which was generated by the Mackey-Glass delay-difference equation. It was used because it has proved to be difficult for

sigmoidal neural networks (Platt, 1991). The application uses 4 inputs and 1 output. The Mackey-Glass time series is representative of time series found in medical applications such as detecting arrhythmias in heartbeats. It could be advantageous to implement this application with portable hardware.

### 4.3 The Speech Phoneme Recognition Application

The speech phoneme recognition problem used the same data as Waibel (Waibel *et al.*, 1989) to learn to recognize the acoustically similar phonemes of b, d and g. The application uses 240 inputs and 3 outputs. The speech phoneme recognition problem represents a sub problem of the more difficult continuous speech recognition problem. Speech recognition applications also represent opportunities for portable computing.

## 5.0 Results

### 5.1 The Remote Sensing Application

Using the RBF neural network on the remote sensing climate problem produced significantly better results than had been previously obtained using conventional statistical methods (Kakar and Lambrigtsen, 1984). The input layer of the RBF network was implemented in two different ways: 1) it was simulated with 32-bit floating point precision to represent a digital input layer, and 2) it was implemented with the analog Gilbert multipliers as the input layer. Both implementations produced similar results.

At an altitude corresponding to 570 mb pressure, the RBF neural network with a digital input layer produced results with .33 absolute rms error vs. .42 rms error for the best results using conventional methods. This is an improvement of 21%. Figure 4 shows the plot of retrieved vs. actual water vapor content for both the RBF network and the conventional method. Using the hybrid neurocomputer with the analog input layer for the data at 570 mb pressure produced results with .338 rms error. This is an improvement of 19.5% over the conventional method. Using the analog input layer produced nearly as much improvement as a completely-digital system, demonstrating the feasibility of placing the network on board a satellite. Similar results were obtained for other altitudes.

The RBF network also was compared to a sigmoidal network using back propagation learning enhanced with line-search capability (to automatically set step-size). Both networks used eight neurons in the hidden layer. As Figure 5 shows, the RBF network learned much faster than the sigmoidal network.

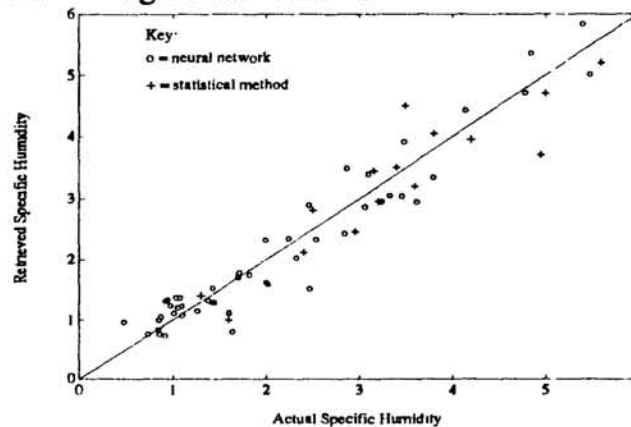

Figure 4: Comparison of Retrieved vs. Actual Water Vapor Content for 570 mb Pressure for RBF Network and Conventional Statistical Method

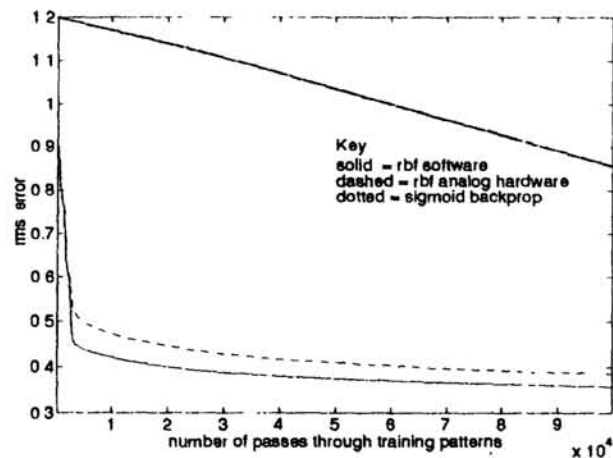

Figure 5: Comparison of Learning Curves for RBF and Sigmoidal Networks for Water Vapor Application

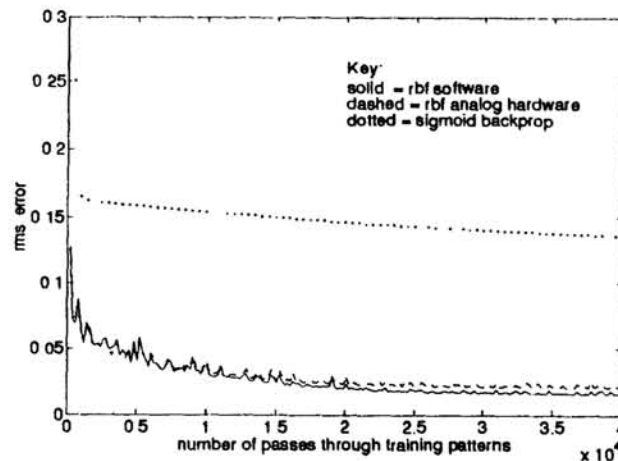

Figure 6: Comparison of Learning Curves for RBF and Sigmoidal Networks for Mackey-Glass Application

## 5.2 The Mackey-Glass Application

The RBF network was not compared to any non-neural network method for the Mackey-Glass time series estimation. It was only compared to a traditional sigmoidal network using back propagation learning enhanced with line search. Both networks used four neurons. As Figure 6 shows, applying the RBF neural network to the Mackey-Glass chaotic time series estimation produced much faster learning than the sigmoidal network. The RBF network with a digital input layer and the RBF hybrid network with an analog input layer both produced similar results in dropping to an rms error of about .025 after only 5 minutes of training on a PC using a 486 CPU.

Using the digital input layer, the RBF network reached a minimum absolute rms error of .017, while the sigmoidal network reached a minimum absolute rms error of .025. This is an improvement of 32% over the sigmoidal network. Using the hybrid neurocomputer with the analog input layer produced a minimum absolute rms error of .022. This is an improvement of 12% over the sigmoidal network

### 5.3 The Speech Phoneme Recognition Application

The RBF network was not compared to any non-neural network method for the speech phoneme recognition problem. It was only compared to Waibel's Time Delay Neural Network (TDNN) (Waibel *et al.*, 1989). The TDNN uses a topology matched to the time-varying nature of speech with two hidden layers of eight and three neurons respectively. The RBF network used a single hidden layer with the number of neurons varying between eight and one hundred.

The TDNN achieved a 98% accuracy on the test set discriminating between the phonemes b, d and g. The RBF network achieved over 99% accuracy in training, but was only able to achieve an 86% accuracy on the test set. To obtain better results, it is clear that the topology of the RBF network needs to be altered to more closely match Waibel's TDNN. However, this topology will complicate the VLSI implementation.

### 5.4 The Feasibility of Using the Analog Input Layer

One potential problem with using an analog input layer is that every individual hybrid RBF neurocomputer might need to be trained on a problem, rather than being able to use a common set of weights obtained from another RBF neurocomputer (which had been previously trained). This potential problem exists because every analog circuit is unique due to variation in the fabrication process. A set of experiments was designed to test this possibility.

The remote sensing application and the Mackey-Glass application were trained and tested two different ways: 1) hardware-trained/hardware-tested, that is, the analog input layer was used for both training and testing; 2) software-trained/hardware-tested, that is the analog input layer was simulated with 32-bit floating point precision for training and then the analog hardware was used for testing. The hardware/hardware results provided a benchmark. The software/hardware results demonstrated the feasibility of having a standard set of weights that are not particular to a given set of analog hardware. For both the remote sensing and the Mackey-Glass applications, the rms error performance was only slightly degraded by using weights learned during software simulation. The remote sensing results degraded by only .011 in terms of absolute rms error, and the Mackey-Glass results degraded by only .002 in terms of absolute rms error. The results of the experiment indicate that each individual hybrid RBF neurocomputer only needs to be calibrated, not trained.

## 6.0 Conclusions

A low-power, hybrid analog/digital neurocomputer development system was constructed using custom hardware. The system implements a radial basis function (RBF) network and is targeted for applications that require low power consumption and use analog data as their input, particularly remote sensing and portable applications. Several applications were executed and results were obtained for a remote sensing application that are superior to any previous results. Comparison of the results of a completely-digital simulation of the RBF network and the hybrid analog/digital RBF network demonstrated the feasibility of the hybrid approach.

## Acknowledgments

The research described in this paper was performed at the Center for Space Microelectronics Technology, Jet Propulsion Laboratory, California Institute of Technology, and was sponsored by the National Aeronautics and Space Administration. One of the authors, Steven S. Watkins, acknowledges the receipt of a Graduate Student Researcher's Center Fellowship from the National Aeronautics and Space Administration. Useful discussions with Silvio Eberhardt, Ron Fellman, Eric Fossum, Doug Kerns, Fernando Pineda, John Platt, and Anil Thakoor are also gratefully acknowledged.

## References

Ramesh Kakar and Bjorn Lambrigtsen, "A Statistical Correlation Method for the Retrieval of Atmospheric Moisture Profiles by Microwave Radiometry," *Journal of Climate and Applied Meteorology*, vol. 23, no. 7, July 1984, pp. 1110-1114.

R. P. Lippman, "A Critical Overview of Neural Network Pattern Classifiers," *Proceedings of the IEEE Neural Networks for Signal Processing Workshop*, 1991, Princeton, N.J., pp. 266-275.

Carver Mead, *Analog VLSI and Neural Systems*, Addison-Wesley, 1989, pp. 90-94.

J. Moody and C. Darken, "Fast Learning in Networks of Locally-Tuned Processing Units," *Neural Computation*, vol. 1, no. 2, Summer 1989, pp. 281-294.

Howard Motteler, J.A. Gualtieri, L.L. Strow and Larry McMillin, "Neural Networks for Atmospheric Retrievals," *NASA Goddard Conference on Space Applications of Artificial Intelligence*, 1993, pp. 155-167.

John Platt, "A Resource-Allocating Neural Network for Function Interpolation," *Neural Computation*, vol. 3, no. 2, Summer 1991, pp. 213-225.

John Platt, Janeen Anderson and David B. Kirk, "An Analog VLSI Chip for Radial Basis Functions," *NIPS 5*, 1993, pp. 765-772.

Alexander Waibel, T. Hanazawa, G. Hinton, K. Shikano and K. Lang, "Phoneme Recognition Using Time-Delay Neural Networks," *IEEE International Conference on Acoustics, Speech and Signal Processing*, May 1989, pp. 393-404.

Steve Watkins, Paul Chau and Raoul Tawel, "A Radial Basis Function Neurocomputer with an Analog Input Layer," *Proceedings of the IJCNN*, Beijing, China, November 1992, pp. III 225-230.

Steve Watkins, Paul Chau and Raoul Tawel, "Different Approaches to Implementing A Radial Basis Function Neurocomputer," *RNNS/IEEE Symposium on Neuroinformatics and Neurocomputing*, Rostov-on-Don, Russia, October 1992, pp. 1149-1155.